# A Bayesian Approach to Concept Drift

**Stephen H. Bach    Marcus A. Maloof**
Department of Computer Science
Georgetown University
Washington, DC 20007, USA
{bach, maloof}@cs.georgetown.edu

## Abstract

To cope with concept drift, we placed a probability distribution over the location of the most-recent drift point. We used Bayesian model comparison to update this distribution from the predictions of models trained on blocks of consecutive observations and pruned potential drift points with low probability. We compare our approach to a non-probabilistic method for drift and a probabilistic method for change-point detection. In our experiments, our approach generally yielded improved accuracy and/or speed over these other methods.

## 1   Introduction

Consider a classification task, in which the objective is to assign labels $Y$ to vectors of one or more attribute values $X$. To learn to perform this task, we use training data to model $f : X \rightarrow Y$, the unknown mapping from attribute values to labels, or *target concept*, in hopes of maximizing classification accuracy. A common problem in online classification tasks is *concept drift*, which is when the target concept changes over time. Identifying concept drift is often difficult. If the correct label for some $x$ is $y_1$ at time step $t_1$ and $y_2$ at time step $t_2$, does this indicate concept drift or that the training examples are noisy?

Researchers have approached drift in a number of ways. Schlimmer and Grainger [1] searched for candidate models by reweighting training examples according to how well they fit future examples. Some have maintained and modified partially learned models, e.g., [2, 3]. Many have maintained and compared "base" models trained on blocks of consecutive examples to identify those that are the best predictors of new examples, e.g., [4, 5, 6, 7, 8]. We focus on this approach. Such methods address directly the uncertainty about the existence and location of drift.

We propose using probability theory to reason about this uncertainty. A probabilistic model of drift offers three main benefits to the research community. First, our experimental results show that a probabilistic model can achieve new combinations of accuracy and speed on classification tasks. Second, probability theory is a well-developed theory that could offer new insights into the problem of concept drift. Third, probabilistic models can easily be combined in a principled way, and their use in the machine-learning field continues to grow [9]. Therefore, our model could readily and correctly share information with other probabilistic models or be incorporated into broader ones.

In this paper we present a probabilistic model of the number of most-recent training examples that the active concept describes. Maximum-likelihood estimation would overfit the model by concluding that each training was generated by a different target concept. This is unhelpful for future predictions, since it eliminates all generalization from past examples to future predictions. Instead, we use Bayesian model comparison [9], or BMC, to reason about the trade-offs between model complexity (i.e., the number of target concepts) and goodness of fit. We first describe BMC and its application to detecting change points. We then describe a Bayesian approach to concept drift. Finally, we show the results of an empirical comparison among our method (pruned and unpruned), BMC for change points, and Dynamic Weighted Majority [5], an ensemble method for concept drift.

## 2 Bayesian model comparison

BMC uses probability theory to assign degrees of belief to candidate models given observations and prior beliefs [9]. By Bayes' Theorem, $p(M|D) = \frac{p(D|M)p(M)}{p(D)}$, where $M$ is the set of models under consideration and $D$ is the set of observations. Researchers in Bayesian statistics have used BMC to look for change points in time-series data. The goal of change-point detection is to segment sequences of observations into blocks that are identically distributed and usually assumed to be independent.

### 2.1 Previous work on Bayesian change-point detection

Barry and Hartigan [10, 11] used product partition models as distributions over possible segmentations of time-series data. Exact inference requires $O(n^3)$ time in the number of observations and may be accurately approximated in $O(n)$ time using Markov sampling [10]. In an online task, approximate training and testing on $n$ observations would require $O(n^2)$ time, since the model must be updated after new training data. These updates would require resampling and testing for convergence.

Fearnhead [12] showed how to perform direct simulation from the posterior distribution of a class of multiple-change-point models. This method requires $O(n^2)$ time and avoids the need to use Markov sampling and to test for convergence. Again, an approximate method can be performed in approximately linear time, but the model must be regularly rebuilt in online tasks.

The computational costs associated with offline methods make it difficult to apply them to online tasks. Researchers have also looked for online methods for change-point detection. Fearnhead and Liu [13] introduced an online version of Fearnhead's simulation method [12] which uses particle filtering to quickly update the distribution over change points. Adams and MacKay [14] proposed an alternative method for online Bayesian change-point detection. We now describe it in more detail, since it will be the starting point for our own model.

### 2.2 A method for online Bayesian change-point detection

Adams and MacKay [14] proposed maintaining a discrete distribution over $l_t$, the length in time steps of the longest substrings of observations that are identically distributed, ending at time step $t$. This method therefore models the location of only the most recent change point, a cost-saving measure useful for many online problems.

A conditional prior distribution $p(l_t|l_{t-1})$ is used, such that

$$p(l_t|l_{t-1}) = \begin{cases} \lambda^{-1} & \text{if } l_t = 0; \\ 1 - \lambda^{-1} & \text{if } l_t = l_{t-1} + 1; \\ 0 & \text{otherwise.} \end{cases} \tag{1}$$

In principle, a more sophisticated prior could be used. The crucial aspect is that, given that a substring is identically distributed, it assigns mass to only two outcomes: the next observation is distributed identically to the observations of the substring, or it is the first of a new substring.

The algorithm is initialized at time step 0 with a single base model that is the prior distribution over observations. Initially, $p(l_0 = 0) = 1$. Let $D_t$ be the observation(s) made at time step $t$. At each time step the algorithm computes a new posterior distribution $p(l_t|D_{1:t})$ by marginalizing out $l_{t-1}$ from

$$p(l_t, l_{t-1}|D_{1:t}) = \frac{p(D_t|l_t, D_{1:t-1})p(l_t|l_{t-1})p(l_{t-1}|D_{1:t-1})}{p(D_t|D_{1:t-1})}. \tag{2}$$

This is a straightforward summation over a discrete variable.

To find $p(l_t, l_{t-1}|D_{1:t})$, consider the three components in the numerator. First, $p(l_{t-1}|D_{1:t-1})$ is the distribution that was calculated at the previous time step. Next, $p(l_t|l_{t-1})$ is the prior distribution. Since only two outcomes are assigned any mass, each element in $p(l_{t-1}|D_{1:t-1})$ contributes mass to only two points in the posterior distribution. This keeps the algorithm linear in the size of the ensemble. Finally, $p(D_t|l_t, D_{1:t-1}) = p(D_t|D_{t-l_t:t-1})$. In other words, it is the predictive probability

of a model trained on the observations received from time steps $t - l_t$ to $t - 1$. The denominator then normalizes the distribution.

Once this posterior distribution $p(l_t|D_{1:t})$ is calculated, each model in the ensemble is trained on the new observation. Then, a new model is initialized with the prior distribution over observations, corresponding to $l_{t+1} = 0$.

## 3    Comparing conditional distributions for concept drift

We propose a new approach to coping with concept drift. Since the objective is to maximize classification accuracy, we want to model the conditional distribution $p(Y|X)$ as accurately as possible. Using [14] as a starting point, we place a distribution over $l_t$, which now refers to the length in time steps that the currently active concept has been active.

There is now an important distinction between BMC for concept drift and BMC for change points: BMC for concept drift models changes in $p(Y|X)$, whereas BMC for change points models changes in the joint distribution $p(Y, X)$. We use the conditional distribution to look for drift points because we do not wish to react to changes in the marginal distribution $p(X)$. A change point in the joint distribution $p(Y, X)$ could correspond to a change point in $p(X)$, a drift point in $p(Y|X)$, or both. Reacting only to changes in $p(Y|X)$ means that we compare models on their ability to classify unlabeled attribute values, not generate those values.

In other words, we assume that neither the sequence of attribute values $X_{1:t}$ nor the sequence of class labels $Y_{1:t}$ alone provide information about $l_t$. Therefore $p(l_t|l_{t-1}, X_t) = p(l_t|l_{t-1})$ and $p(l_{t-1}|Y_{1:t-1}, X_{1:t}) = p(l_{t-1}|Y_{1:t-1}, X_{1:t-1})$. We also assume that examples from different concepts are independent. We use Equation 1 as the prior distribution $p(l_t|l_{t-1})$ [14]. Equation 2 is replaced with

$$p(l_t, l_{t-1}|Y_{1:t}, X_{1:t}) = \frac{p(Y_t|l_t, Y_{1:t-1}, X_{1:t})p(l_t|l_{t-1})p(l_{t-1}|Y_{1:t-1}, X_{1:t-1})}{p(Y_t|Y_{1:t-1}, X_{1:t})}. \qquad (3)$$

To classify unlabeled attribute values $X$ with class label $Y$, the predictive distribution is

$$p(Y|X) = \sum_{i=1}^{t} p(Y|X, Y_{1:t}, X_{1:t}, l_t = i)p(l_t = i). \qquad (4)$$

We call this method Bayesian Conditional Model Comparison (BCMC). If left unchecked, the size of its ensemble will grow linearly with the number of observations. In practice, this is far too computationally expensive for many online-learning tasks. We therefore prune the set of models during learning. Let $\phi$ be a user-specified threshold for the minimum posterior probability a model must have to remain in the ensemble. Then, if there exists some $i$ such that $p(l_t = i|D_{1:t}) < \phi < p(l_t = 0|l_{t-1})$, simply set $p(l_t = i|D_t) = 0$ and discard the model $p(D|D_{t-i:t})$. We call this modified method Pruned Bayesian Conditional Model Comparison (PBCMC).

## 4    Experiments

We conducted an empirical comparison using our implementations of PBCMC and BCMC. We hypothesized that looking for drift points in the conditional distribution $p(Y|X)$ instead of change points in the joint distribution $p(Y, X)$ would lead to higher accuracy on classification tasks. To test this, we included our implementation of the method of Adams and MacKay [14], which we refer to simply as BMC. It is identical to BCMC, except that it uses Equation 2 to compute the posterior over $l_t$, where $D \equiv (Y, X)$.

We also hypothesized that PBCMC could achieve improved combinations of accuracy and speed compared to Dynamic Weighted Majority (DWM) [5], an ensemble method for concept drift that uses a heuristic weighting scheme and pruning. DWM is a top performer on the problems we considered [5]. Like the other learners, DWM maintains a dynamically-sized, weighted ensemble of models trained on blocks of examples. It predicts by taking a weighted-majority vote of the models' predictions and multiplies the weights of those models that predict incorrectly by a constant $\beta$. It

then rescales the weights so that maximum weight is 1. Then if the algorithm's global prediction was incorrect, it adds a new model to the ensemble with a weight of 1, and it removes any models with weights below a threshold $\theta$. In the cases of models which output probabilities, DWM considers a prediction incorrect if a model did not assign the most probability to the correct label.

## 4.1 Test problems

We conducted our experiments using four problems previously used in the literature to evaluate methods for concept drift The STAGGER concepts [1, 3] are three target concepts in a binary classification task presented over 120 time steps. Attributes and their possible values are *shape* $\in$ {*triangle, circle, rectangle*}, *color* $\in$ {*red, green, blue*}, and *size* $\in$ {*small, medium, large*}. For the first 40 time steps, the target concept is *color = red $\land$ size = small*. For the next 40 time steps, the target concept is *color = green $\lor$ shape = circle*. Finally, for the last 40 time steps, the target concept is *size = medium $\lor$ size = large*. A number of researchers have used this problem to evaluate methods for concept drift [4, 5, 3, 1]. Per the problem's usual formulation, we evaluated each learner by presenting it with a single, random example at each time step and then testing it on a set of 100 random examples, resampled after each time step. We conducted 50 trials.

The SEA concepts [8] are four target concepts in a binary classification task, presented over 50,000 time steps. The target concept changes every 12,500 time steps, and associated with each concept is a single, randomly generated test set of 2,500 examples. At each time step, a learner is presented with a randomly generated example, which has a 10% chance of being labeled as the wrong class. Every 100 time steps, the learner is tested on the active concept's test set. Each example consists of numeric attributes $x_i \in [0, 10]$, for $i = 1, \ldots, 3$. The target concepts are hyperplanes, such that $y = +$ if $x_1 + x_2 \leq \theta$, where $\theta \in \{7, 8, 9, 9.5\}$, for each of the four target concepts, respectively; otherwise, $y = -$. Note that $x_3$ is an irrelevant attribute. Several researchers have used a shifting hyperplane to evaluate learners for concept drift [5, 6, 7, 2, 8]. We conducted 10 trials. In this experiment, $\mu_0 = 5$.

The calendar-apprentice (CAP) data sets [15, 16] is a personal-scheduling task. Using a subset of 34 symbolic attributes, the task is to predict a user's preference for a meeting's location, duration, start time, and day of week. There are 12 attributes for location, 11 for duration, 15 for start time, and 16 for day of week. Each learner was tested on the 1,685 examples for User 1. At each time step, the learner was presented the next example without its label. After classifying it, it was then told the correct label so it could learn.

The electricity-prediction data set consists of 45,312 examples collected at 30-minute intervals between 7 May 1996 and 5 December 1998 [17]. The task is to predict whether the price of electricity will go up or down based on five numeric attributes: the day of the week, the 30-minute period of the day, the demand for electricity in New South Wales, the demand in Victoria, and the amount of electricity to be transferred between the two. About 39% of the examples have unknown values for either demand in Victoria or the transfer amount. At each time step, the learner classified the next example in temporal order before being given the correct label and using it to learn. In this experiment, $\mu_0 = 0$.

## 4.2 Experimental design

We tested the learning methods on the four problems described. For STAGGER and SEA, we measured accuracy on the test set, then computed average accuracy and 95% confidence intervals at each time step. We also computed the average normalized area under the performance curves (AUC) with 95% confidence intervals. We used the trapezoid rule on adjacent pairs of accuracies and normalized by dividing by the total area of the region. We present both AUC under the entire curve and after the first drift point to show both a learner's overall performance and its performance after drift occurs. For CAP and electricity prediction, we measured accuracy on the unlabeled observations.

All the learning methods used a model we call Bayesian Naive Bayes, or BNB, as their base models. BNB makes the conditionally independent factor assumption (a.k.a. the "naive Bayes" assumption) that the joint distribution $p(Y, X)$ factors into $p(Y) \prod_{i=1}^{n} p(X_i|Y)$ [9]. It calculates values for $p(Y|X)$ as needed using Bayes' Theorem. It takes the Bayesian approach to probabilities (hence the additional "Bayes" in the name), meaning that it places distributions over the parameters that govern

Table 1: Results for (a) the STAGGER concepts and (b) the SEA concepts.

(a) STAGGER concepts

| Learner and Parameters | AUC (overall) | AUC (after drift) |
|---|---|---|
| BNB, on each concept | 0.912±0.005 | 0.914±0.007 |
| PBCMC, $\lambda = 20, \phi = 10^{-4}$ | 0.891±0.005 | 0.885±0.007 |
| BCMC, $\lambda = 20$ | 0.891±0.005 | 0.885±0.007 |
| BMC, $\lambda = 50$ | 0.884±0.005 | 0.876±0.008 |
| DWM, $\beta = 0.5, \theta = 10^{-4}$ | 0.878±0.005 | 0.868±0.007 |
| BNB, on all examples | 0.647±0.008 | 0.516±0.011 |

(b) SEA concepts

| Learner and Parameters | AUC (overall) | AUC (after drift) |
|---|---|---|
| BNB, on each concept | 0.974±0.002 | 0.974±0.002 |
| DWM, $\beta = 0.9, \theta = 10^{-3}$ | 0.974±0.001 | 0.974±0.001 |
| BCMC, $\lambda = 10,000$ | 0.970±0.002 | 0.969±0.002 |
| PBCMC, $\lambda = 10,000, \phi = 10^{-4}$ | 0.964±0.002 | 0.961±0.003 |
| BMC, $\lambda = 200$ | 0.955±0.003 | 0.948±0.003 |
| BNB, on all examples | 0.910±0.003 | 0.889±0.002 |

the distributions $p(Y)$ and $p(X|Y)$ into which $p(Y,X)$ factors. In our experiments, BNB predicted by marginalizing out the latent parameter variables to compute marginal likelihoods. Note that we use BNB, a generative model over $p(Y,X)$, even though we said that we wish to model $p(Y|X)$ as accurately as possible. This is to ensure a fair comparison with BMC which needs $p(Y,X)$. We are more interested in the effects of looking for changes in each distribution, not which is a better model for the active concept.

In our experiments, BNB placed Dirichlet distributions [9] over the parameters $\vec{\theta}$ of multinomial distributions $p(Y)$ and $p(X_i|Y)$ when $X_i$ was a discrete attribute. All Dirichlet priors assigned equal density to all valid values of $\vec{\theta}$. BNB placed Normal-Gamma distributions [9] over the parameters $\mu$ and $\lambda$ of normal distributions $p(X_i|Y)$ when $X_i$ was a continuous attribute. $p(\mu, \lambda) = \mathcal{N}(\mu|\mu_0, (\beta\lambda)^{-1})\text{Gam}(\lambda|a,b)$. The predictive distribution is then a Student's t-distribution with mean $\mu$ and precision $\lambda$. In all of our experiments, $\beta = 2$ and $a = b = 1$. The value of $\mu_0$ is specified for each experiment with continuous attributes.

We also tested BNB as a control to show the effects of not attempting to cope with drift and BNB trained using only examples from the active concept (when such information was available) to show possible accuracy given perfect information about drift.

Parameter selection is difficult when evaluating methods for concept drift. Train-test-and-validate methods such as $k$-fold cross validation are not appropriate because the observations are ordered and not assumed to be identically distributed. We therefore tested each learner on each problem using each of a set of values for each parameter. Due to limited space, we present results for each learning method using the best parameter settings we found. We make no claim that these parameters are optimal, but they are representative of the overall trends we observed. We performed this parameter search for all the learning methods. The parameters we tested were $\lambda \in \{10, 20, 50, 100, 200, 500, 1000, 2000, 5000, 10000\}$, $\phi \in \{10^{-2}, 10^{-3}, 10^{-4}\}$, $\beta \in \{0.25, 0.5, 0.75, 0.9\}$, and $\theta \in \{10^{-2}, 10^{-3}, 10^{-4}, 0\}$.

Table 2: Accuracy on the CAP and electricity data sets.

| | PBCMC | BCMC | BMC | DWM | BNB |
|---|---|---|---|---|---|
| | $\lambda = 10{,}000, \phi = 10^{-4}$ | $\lambda = 5{,}000$ | $\lambda = 10$ | $\beta = 0.75, \theta = 10^{-4}$ | |
| Location | 63.74 | 63.92 | 63.15 | **65.76** | 62.14 |
| Duration | 63.15 | 63.03 | 64.10 | **66.35** | 62.37 |
| Start Time | 38.40 | **39.17** | 35.19 | 37.98 | 32.40 |
| Day of Week | **51.81** | **51.81** | 51.22 | 51.28 | 51.22 |
| Average | 54.27 | 54.48 | 53.41 | **55.34** | 52.03 |
| | $\lambda = 10, \phi = 10^{-2}$ | $\lambda = 10$ | $\lambda = 10$ | $\beta = 0.25, \theta = 10^{-3}$ | |
| Electricity | 85.32 | **85.33** | 65.37 | 82.31 | 62.44 |

## 4.3 Results and analysis

Table 1 shows the top results for the STAGGER and SEA concepts. On the STAGGER concepts, PBCMC and BCMC performed almost identically and have a higher mean AUC than BMC, but their 95% confidence intervals overlap. PBCMC and BCMC outperformed DWM. On the SEA concepts, DWM was the top performer, matching the accuracy of BNB trained on each concept and outperforming all the other learner methods. BCMC was next, followed by PBCMC, then BMC, and the BNB.

Table 2 shows the top results for the CAP and electricity data sets. DWM performed the best on the location and duration data sets, while BCMC performed best on the start time and day-of-week data sets. PBCMC matched the accuracy of BCMC on the day-of-week and duration data sets and came close to it on the others. DWM had the highest mean accuracy over all four tasks, followed by PBCMC and BCMC, then BMC, and finally BNB. BCMC performed the best on the electricity data set, closely followed by PBCMC.

The first conclusion is clear: looking for changes in the conditional distribution $p(Y|X)$ led to better accuracy than looking for changes in the joint distribution $p(Y, X)$. With the close exception of the duration problem in the CAP data sets, PBCMC and BCMC outperformed BMC, sometimes dramatically so. What is less clear is the relative merits of PBCMC and DWM. We now analyze these learners to better understand address this question.

### 4.3.1 Reactivity versus stability

The four test problems can be partitioned into two subsets: those on which PBCMC was generally more accurate (STAGGER and electricity) and those on which DWM was (SEA and CAP). We can obtain further insight into what separates these two subsets by noting that both PBCMC and DWM can be said to have "strategies," which are determined by their parameters. For PBCMC, higher values of $\lambda$ mean that it will assign less probability initially to new models. For DWM, higher values of $\beta$ mean that it will penalize models less for making mistakes. For both, lower values of $\phi$ and $\theta$ respectively mean that they are slower to completely remove poorly performing models from consideration. We can thus interpret these parameters to describe how "reactive" or "stable" the learners are, i.e., the degree to which new observations can alter their hypotheses [4].

The two subsets are also partitioned by the strategy which was superior for the problems in each. For both PBCMC and DWM, some of the most reactive parameterizations we tested were optimal on STAGGER and electricity, but some of the most stable were optimal on SEA and CAP. Further, we observed generally stratified results across parameterizations. For each problem, almost all of the parameterizations of the top learner were more accurate than almost all of the parameterizations of the other. This indicates that PBCMC was generally better for the concepts which favor reactivity, whereas DWM was generally better for the concepts which favor stability.

### 4.3.2 Closing the performance gaps

We now consider why these gaps in performance exist and how they might be closed. Figure 1 shows the average accuracies of PBCMC and DWM at each time step on the STAGGER and SEA concepts. These are for the experiments reported in Table 1, so the parameters, numbers of trials, etc. are the same. We present 95% confidence intervals at selected time steps for both. Figure 1 shows that the

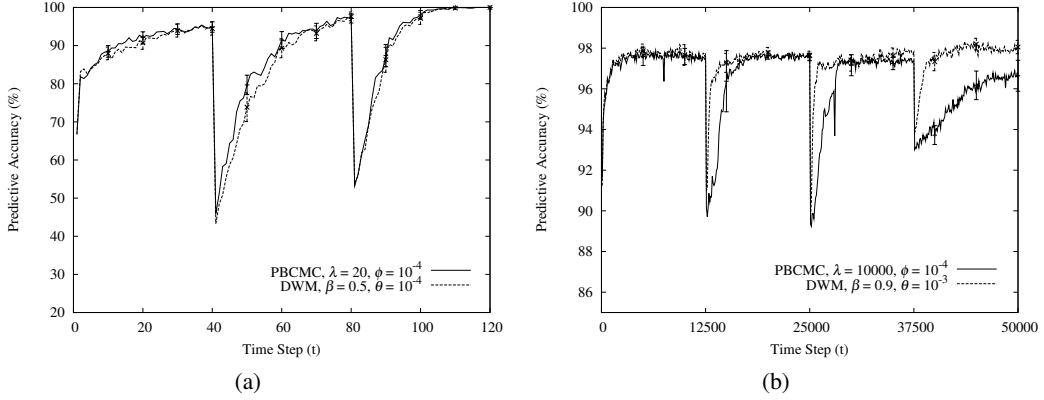

Figure 1: Average accuracy on (a) the STAGGER concepts and (b) the SEA concepts. See text for details.

better performing learners in each problem were faster to react to concept drift. This shows that DWM did not perform better on SEA simply by being more stable whether the concept was or not. On the SEA concepts, PBCMC did perform best with the most stable parameterization we tried, but its main problem was that it wasn't *reactive* enough *when* drift occurred.

We first consider whether the problem is one of parameter selection. Perhaps we can achieve better performances by using a more reactive parameterization of DWM on certain problems and/or a more stable parameterization of PBCMC on other problems. Our experimental results cast doubt on this proposition. For the problems on which PBCMC was superior, DWM's best results were not obtained using *the* most reactive parameterization. In other words, simply using an even more reactive parameterization of DWM did not improve performance on these problems. Further, on the duration problem in the CAP data sets, PBCMC also achieved the reported accuracy using $\lambda = 5000$ and $\phi = 10^{-2}$, and on the location problem it acheived negligibly better accuracy using $\lambda = 5000$ and $\phi = 10^{-3}$ or $\phi = 10^{-4}$. Therefore, simply using an even more stable parameterization of PBCMC did not improve performance on these problems either. BCMC, which is just PBCMC with $\phi = 0$, did outperform PBCMC on SEA. It reacted more quickly than PBCMC did, but not as quickly as DWM did, and at a much greater computational cost, since it had to maintain every model in order to have the one(s) which would eventually gain weight relative to the other models. BCMC also was not a significant improvement over PBCMC on the location and duration problems.

We therefore theorize that the primary reason for the differences in performance between PBCMC and DWM is their approaches to updating their ensembles, which determines how they react to drift. PBCMC favors reactivity by adding a new model at every time step and decaying the weights of all models by the degree to which they are incorrect. DWM favors stability by only adding a new model after incorrect overall predictions and only decaying weights of incorrect models, and then only by a constant factor. This is supported by the results on problems favoring reactive parameterizations compared with the results on problems favoring stable parameterizations. Further, that it is difficult to close the performance gaps with better parameter selection suggests that there is a range of reactivity or stability each favors. When parameterized beyond this range, the performance of each learner degrades, or at least plateaus.

To further support this theory, we consider trends in ensemble sizes. Figure 2 shows the average number of models in the ensembles of PBCMC and DWM at each time step on the STAGGER and SEA concepts. These are again for the experiments reported in Table 1, and again we present 95% confidence intervals at selected time steps for both. The figure shows that the trends in ensemble sizes were roughly interchanged between the two learners on the two problems. On both problems, one learner stayed within a relatively small range of ensemble sizes, whereas the other continued to expand the ensemble when the concept was stable, only significantly pruning soon after drift. On STAGGER, PBCMC expanded its ensemble size far more, whereas DWM did on SEA. This agrees with our expectations for the synthetic concepts. STAGGER contains no noise, whereas SEA does, which complements the designs of the two learners. When noise is more likely, DWM will update

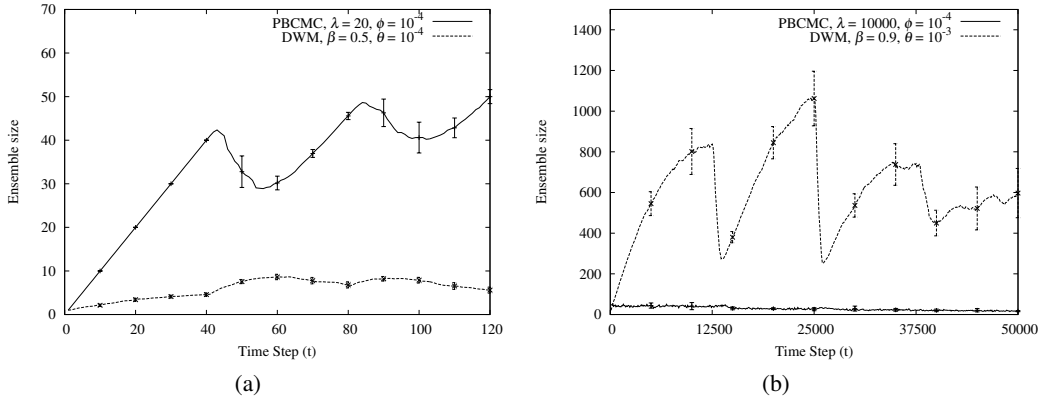

Figure 2: Average numbers of models on (a) the STAGGER concepts and (b) the SEA concepts. See text for details.

its ensemble more than when it is not as likely. However, when noise is more likely, PBCMC will usually have difficulty preserving high weights for models which are actually useful. Conversely, PBCMC regularly updates its ensemble, and DWM will have less difficulty maintaining high weights on good models because it only decays weights by a constant factor.

Therefore, it seems that each learner reaches the boundary of its favored range of reactivity or stability when further changes in that direction cause it to either be so reactive that it often assigns relatively high probability of drift to many time steps for which there was no drift, or so stable that it cannot react to actual drift. On STAGGER, PBCMC matched the performance of BNB on the first target concept (not shown), whereas DWM made more mistakes as it reacted to erroneously inferred drift. On SEA, PBCMC needs to be parameterized to be so stable that it cannot react quickly to drift.

# 5 Conclusion and Future Work

In this paper we presented a Bayesian approach to coping with concept drift. Empirical evaluations supported our method. We showed that looking for changes in the conditional distribution $p(Y|X)$ led to better accuracy than looking for changes in the joint distribution $p(Y, X)$. We also showed that our Bayesian approach is competitive with one of the top ensemble methods for concept drift, DWM, sometimes beating and sometimes losing to it. Finally, we explored why each method sometimes outperforms the other. We showed that both PBCMC and DWM appear to favor a different range of reactivity or stability.

Directions for future work include integrating the advantages of both PBCMC and DWM into a single learner. Related to this task is a better characterization of their relative advantages and the relationships among them, their favored ranges of reactivity or stability, and the problems to which they are applied. It also important to note that the more constrained ensemble sizes discussed above correspond to faster classification speeds. Future work could explore how to balance this desiderata with the desire for better accuracy. Finally, another direction is to integrate a Bayesian approach with other probabilistic models. With a useful probabilistic model for concept drift, such as ours, one could potentially incorporate existing probabilistic domain knowledge to guide the search for drift points or build broader models that use beliefs about drift to guide decision making.

### Acknowledgments

The authors wish to thank the anonymous reviewers for their constructive feedback. The authors also wish to thank Lise Getoor and the Department of Computer Science at the University of Maryland, College Park. This work was supported by the Georgetown University Undergraduate Research Opportunities Program.

# References

[1] J. C. Schlimmer and R. H. Granger. Beyond incremental processing: Tracking concept drift. In *Proceedings of the Fifth National Conference on Artificial Intelligence*, pages 502–507, Menlo Park, CA, 1986. AAAI Press.

[2] G. Hulten, L. Spencer, and P. Domingos. Mining time-changing data streams. In *Proceedings of the Seventh ACM SIGKDD International Conference on Knowledge Discovery and Data Mining*, pages 97–106, New York, NY, 2001. ACM Press.

[3] G. Widmer and M. Kubat. Learning in the presence of concept drift and hidden contexts. *Machine Learning*, 23:69–101, 1996.

[4] S. H. Bach and M. A. Maloof. Paired learners for concept drift. In *Proceedings of the Eighth IEEE International Conference on Data Mining*, pages 23–32, Los Alamitos, CA, 2008. IEEE Press.

[5] J. Z. Kolter and M. A. Maloof. Dynamic weighted majority: An ensemble method for drifting concepts. *Journal of Machine Learning Research*, 8:2755–2790, Dec 2007.

[6] J. Z. Kolter and M. A. Maloof. Using additive expert ensembles to cope with concept drift. In *Proceedings of the Twenty-second International Conference on Machine Learning*, pages 449–456, New York, NY, 2005. ACM Press.

[7] H. Wang, W. Fan, P. S. Yu, and J. Han. Mining concept-drifting data streams using ensemble classifiers. In *Proceedings of the Ninth ACM SIGKDD International Conference on Knowledge Discovery and Data Mining*, pages 226–235, New York, NY, 2003. ACM Press.

[8] W. N. Street and Y. Kim. A streaming ensemble algorithm (SEA) for large-scale classification. In *Proceedings of the Seventh ACM SIGKDD International Conference on Knowledge Discovery and Data Mining*, pages 377–382, New York, NY, 2001. ACM Press.

[9] C. M. Bishop. *Pattern Recognition and Machine Learning*. Springer, Berlin-Heidelberg, 2006.

[10] D. Barry and J. A. Hartigan. A Bayesian analysis for change point problems. *Journal of the American Statistical Association*, 88(421):309–319, 1993.

[11] D. Barry and J. A. Hartigan. Product partition models for change point problems. *The Annals of Statistics*, 20(1):260–279, 1992.

[12] Paul Fearnhead. Exact and efficient Bayesian inference for multiple changepoint problems. *Statistics and Computing*, 16(2):203–213, 2006.

[13] P. Fearnhead and Z. Liu. On-line inference for multiple changepoint problems. *Journal of the Royal Statistical Society: Series B (Statistical Methodology)*, 69(4):589–605, September 2007.

[14] R.P. Adams and D.J.C. MacKay. Bayesian online changepoint detection. Technical report, University of Cambridge, 2007. http://www.inference.phy.cam.ac.uk/rpa23/papers/rpa-changepoint.pdf.

[15] A. Blum. Empirical support for winnow and weighted-majority algorithms: Results on a calendar scheduling domain. *Machine Learning*, 26:5–23, 1997.

[16] T. M. Mitchell, R. Caruana, D. Freitag, J. McDermott, and D. Zabowski. Experience with a learning personal assistant. *Communications of the ACM*, 37(7):80–91, July 1994.

[17] M. Harries, C. Sammut, and K. Horn. Extracting hidden context. *Machine Learning*, 32(2):101–126, 1998.

